# Eye micro-movements improve stimulus detection beyond the Nyquist limit in the peripheral retina

**Matthias H. Hennig and Florentin Wörgötter**
Computational Neuroscience
Psychology
University of Stirling
FK9 4LR Stirling, UK
{hennig,worgott}@cn.stir.ac.uk

## Abstract

Even under perfect fixation the human eye is under steady motion (tremor, microsaccades, slow drift). The "dynamic" theory of vision [1, 2] states that eye-movements can improve hyperacuity. According to this theory, eye movements are thought to create variable spatial excitation patterns on the photoreceptor grid, which will allow for better spatiotemporal summation at later stages. We reexamine this theory using a realistic model of the vertebrate retina by comparing responses of a resting and a moving eye. The performance of simulated ganglion cells in a hyperacuity task is evaluated by ideal observer analysis. We find that in the central retina eye-micromovements have no effect on the performance. Here optical blurring limits vernier acuity. In the retinal periphery however, eye-micromovements clearly improve performance. Based on ROC analysis, our predictions are quantitatively testable in electrophysiological and psychophysical experiments.

## 1 Introduction

Normal visual acuity is limited by the photoreceptor distance on the retina to about $1'$ of visual angle, which is imposed by the neural nyquist sampling limit. The human visual system, however, is capable of resolving certain stimuli (e.g. vernier stimuli) at a much higher resolution of $< 5''$. This effect, called hyperacuity, has given rise to a large number of psychophysical studies and several qualitative theories about perception as well as the underlying neuronal properties. Most notably are the so-called "dynamic" and "static" theories of vision [3], which claim that hyperacuity would require eye-micromovements (microtremor, microsaccades) or not. Along the dynamic theory it has been suggested by Averill and Weymouth [1] and later by Marshall and Talbot [2] that small eye-movements would shift the photoreceptor grid across the stimulus leading to a better discriminability when appropriate spatiotemporal integration is used.

In a previous study we had designed a realistic and detailed model of the vertebrate retina [4]. This allows us for the first time to quantitatively test the Marshall-Talbot

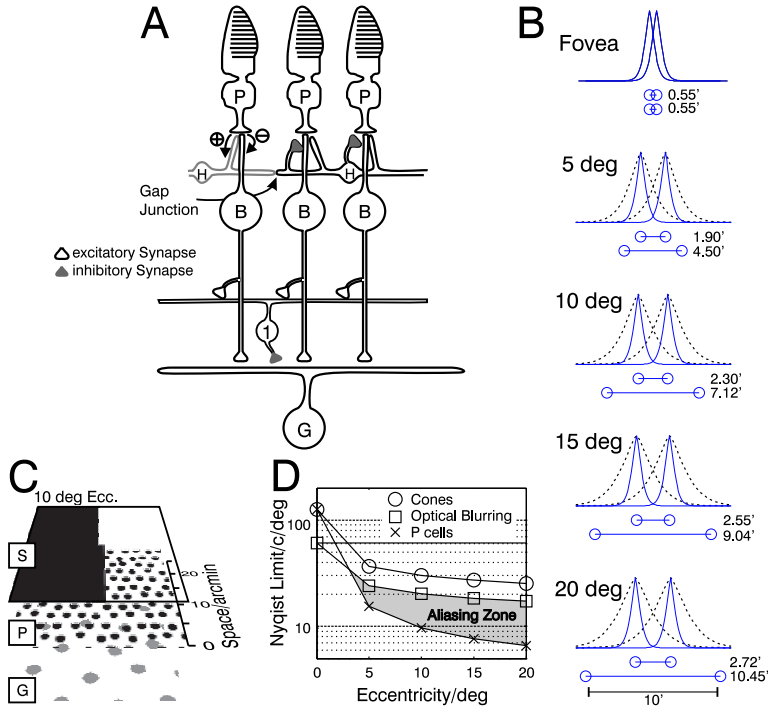

Figure 1: Overview of the model. A, Structure of the retina model. Photoreceptors (P) connect to horizontal (H) and bipolar cells (B). Horizontal cells antagonize bipolar cells. Bipolar cells provide the center input to ganglion cells (G) and the surround is mediated by a Type 1 (1) amacrine cell [4]. B, Scaling of optical point spread functions (top curves), photoreceptor (upper lines, values shown, data from [5]) and ganglion cell separation (lower lines, values shown, data from [6, 7]) at different retinal eccentricities. PSF's are shown for the constant (straight lines) and scaled case (dashed lines). C, Spatial layout of the stimulus (S) and the photoreceptor (P) and ganglion cell (G) grids. D, Nyquist frequencies for photoreceptors, P ganglion cells and the scaled PSF as a function of the eccentricity. Aliasing occurs in the shaded region.

theory under different experimental conditions. We will show that the presence of eye-micromovements indeed improves hyperacuity. Contrary to earlier assumptions we find that eye micromovements have no effect in the central part of the retina, where optical blurring defines the limit for hyperacuity tasks. At above $5°$ retinal eccentricity, eye-micromovements are clearly improving hyperacuity. Our approach relies on a model free (receiver-operator characteristic, ROC) analysis, and the reported results should be directly measurable in retinal ganglion cells and psychophysically.

## 2   MATERIALS AND METHODS

The model used in this study is based on a previously described model of the light adapted retina. In this section, we only mention aspects which are important in the context of this study. For a detailed discussion of the model, see [4].

Briefly, the model consists of cone photoreceptors, horizontal and bipolar, amacrine and ganglion cells (Fig. 1A). Neurons are arranged on homogeneous two-dimensional hexago-

nal grids (Fig. 1C). Ganglion cells are shifted randomly by $12\%$ of their separation to account for the non-ideal distribution on the hexagonal grid. Cones, bipolar and ganglion cells form the feed-forward path and horizontal and amacrine cells two lateral layers. Densities and receptive field sizes of photoreceptors and ganglion cells were adjusted to the anatomical data available for the human retina at the different eccentricities studied (Fig.1B). The separation of horizontal, bipolar and amacrine cells was scaled proportional to the cone density.

| Eccentricity [deg] | PSF scaling | Vernier offset [arcsec] |
|---|---|---|
| 0 | 1.00 | 7 |
| 5 | 2.51 | 46 |
| 10 | 2.98 | 83 |
| 15 | 3.31 | 92 |
| 20 | 3.52 | 98 |

Table 1: Spatial scaling of the PSF that simulates the optical blurring and of the vernier offset as a function of the eccentricity.

The photoreceptor model is a slightly modified version of the mathematical description given by Hennig et al. [4]. It is originally based on a description by Schnapf et al. [8]. The voltage responses were tested against experimental data from the macaque monkey by Schneeweis and Schnapf [9]. To account for the sustained responses for strong, but brief stimuli, the single initial activation stage [4] was replaced by three cascaded low-pass filters. This study focuses on human P On-center cells (or "midget" cells). Receptive field sizes and densities were chosen according to anatomical data (Fig. 1). The center and surround input of both cell types is weighted by overlapping Gaussian profiles [10], where the surround extends $> 3.8$ times the center input [11].

Ocular optical blurring has been accounted for by convolving the stimulus with the point-spread function (PSF) given by Westheimer et al. [12] for the fovea:

$$PSF(\rho) = 0.933 \cdot e^{-2.59 \cdot \rho^{1.36}} + 0.047 \cdot e^{-2.34 \cdot \rho^{1.74}} \qquad (1)$$

$\rho$ is the radius in arcmin. For higher eccentricities two sets of simulations were performed, one with a constant and one with a scaled PSF (Fig.1B). The first case is an approximation of the case when off-axis refractory errors of the ocular optics are corrected. Then aliasing occurs already at the level of the cone mosaic. The more realistic case corresponds to a scaled PSF because off-axis astigmatism and increasing cone aperture increase the amount of blurring at higher eccentricities. Scaling factors were chosen to fit experimental data (Tab. 1, [13]). Under these conditions, aliasing on the ganglion cell layer begins at $5°$ (Fig.1D).

Eye micromovements where modeled by shifting the retina randomly relative to the stimulus by using a data fit by Eizenman et al. (Fig. 2A,B, [14]). They include the ocular microtremor and fast and slow microsaccades (Fig. 2B). Two types of micromovements were used in the simulations in this work: slow and fast microsaccades and the microtremor (MT) and only fast microsaccades and the tremor (FMT).

A typical vernier stimulus has been used in the simulations. To remove the effect of the stimulus size, we used a bipartite field of $100\%$ contrast with a small horizontal displacement in the vertical half (Fig.1C). Simulations were carried out at five different retinal eccentricities: in the fovea and at 5, 10, 15 and 20 deg. The vernier offset was scaled with increasing eccentricity proportional to the ratio of the cone to ganglion cell separation (Tab.1).

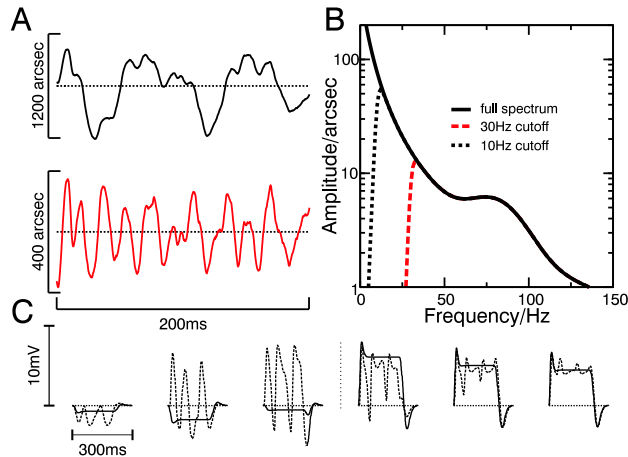

Figure 2: Characteristics of the simulated eye-micromovements. A, Traces of the horizontal retinal displacement for the two tremor spectra used (top: MT, bottom: FMT, see Methods). B, Power spectra of the two cases from part A (dashed line: MT, dotted line: FMT) and the full spectrum given by Eigenman et al. (straight line, [14]). C, Responses of P-ganglion cells to a contrast step (100% contrast) without tremor (solid line) and with eye micromovements (MT, dotted line). Horizontal alignment corresponds to the location of the cell relative to the stimulus (location of contrast step indicated by dotted line).

## 3   Results

Fig. 2 summarizes the characteristics of simulated eye-micromovements. In part A an example for the horizontal displacement of the retina is shown for the two types of micromovements included in the model (MT and FMT, see Methods). Part B shows the corresponding power spectra. Fig. 2C shows the membrane potential of a simulated ganglion cell at different locations relative to a contrast step with and without eye micromovements. When the cell is located in the dark half of the contrast step, moving the light half of the stimulus into its receptive field causes frequent strong depolarizations. For the reverse case, when the dark half of the stimulus moves into the receptive field of a cell which was previously excited, the membrane potential hyperpolarizes. These hyperpolarizations are weaker than the depolarizations in the former case because the photoreceptor response is asymmetric with respect to the to on- and offset of light. Light onset leads a to brief, strong transient hyperpolarization whereas offset causes a slower response decay and a weaker phasic depolarization [4, 9].

Fig. 3A,E show the spatial response distribution on the ganglion cell layer 30ms after stimulus onset for two retinal eccentricities for the constant PSF. At $5°$ eccentricity the vernier offset is well visible by eye by comparing the upper and lower half of the responses. At $10°$ however, upper and lower half look very similar, implying that vernier detection is not possible.

To quantify the detectability of a vernier stimulus we performed a ROC analysis of the spatial response profiles. This procedure is shown in Fig.3: First a horizontal cross-section of the spatial response profile on the ganglion cell layer is taken for the upper and lower part of the stimulus (B, F). The detectability of a vernier stimulus should be reflected in the population average of the ganglion cell responses for upper and lower part of the stimulus. This assumption reflects the known convergence properties of the primary visual pathway, where each cortical cell receives input (via the LGN) from many ganglion cells. We used

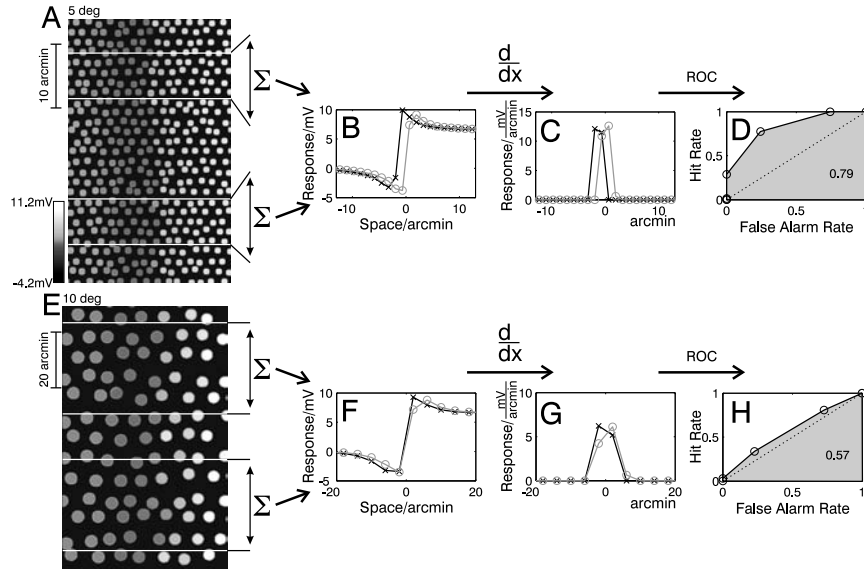

Figure 3: Spatial analysis of the vernier stimuli. A, Spatial response profiles of the ganglion cells to a vernier stimulus 30ms after stimulus onset (5° retinal eccentricity, vernier offset $45''$). The membrane potential is coded by gray levels. B, Spatial response profile for the upper (black) and lower half (grey) of the responses in A (average over four rows). C, Spatial derivative of the curves in B, rectified at zero. D, ROC curve calculated from the curves in C. Value of the integral of the ROC curve (shaded gray) is shown for each curve (detectability index). E-H The same analysis at 10° retinal eccentricity and a vernier offset of $92''$.

an average of four rows of the ganglion cells for analysis. The resulting profiles closely fit cumulative Difference of Gaussians functions, which is a consequence of the ganglion cell receptive field structure. In the next step, the spatial derivative of the response profile is calculated and rectified at the resting potential (C, G). This operation is similar to a cortical edge detection mechanism [15] and leads to Gaussian-like distributions. From these curves it is possible to directly compute a ROC-curve (D, H). The integral of the ROC curve, ranging from 0.5 to 1, is then taken as a direct measure of the detectability of the vernier offset. This method combines the standard, model-free ROC-type analysis with basic assumptions about the convergence properties in the primary visual pathway.

Eye-movements lead to temporal changes of the detectability. Thus, the integral of the ROC curve, which we will call the "detectability index" ($DI$), then varies over time. Fig. 4A shows this effect for the five different retinal eccentricities studied and different types of micromovements using the scaled PSF. For each eccentricity, the stimulus has been placed at five different locations relative to the ganglion cell receptive fields. We found that, without eye-micromovements and increasing eccentricities, the detectability strongly depends on the location of the stimulus in the receptive field. This is not surprising when one considers that spatial undersampling of the stimulus occurs at the ganglion cell layer. At the fovea visual resolution is limited by the optics of the eye. At $> 5°$ eccentricity, there are substantial "gaps" in the ganglion cell representation of the stimulus (see Fig.1B) which cause aliasing effects. Aliasing effects in the periphery due to undersampling has been reported in psychophysics [16].

Ocular micromovements leads to clearly visible effects (Fig. 4A). The noisy curves are

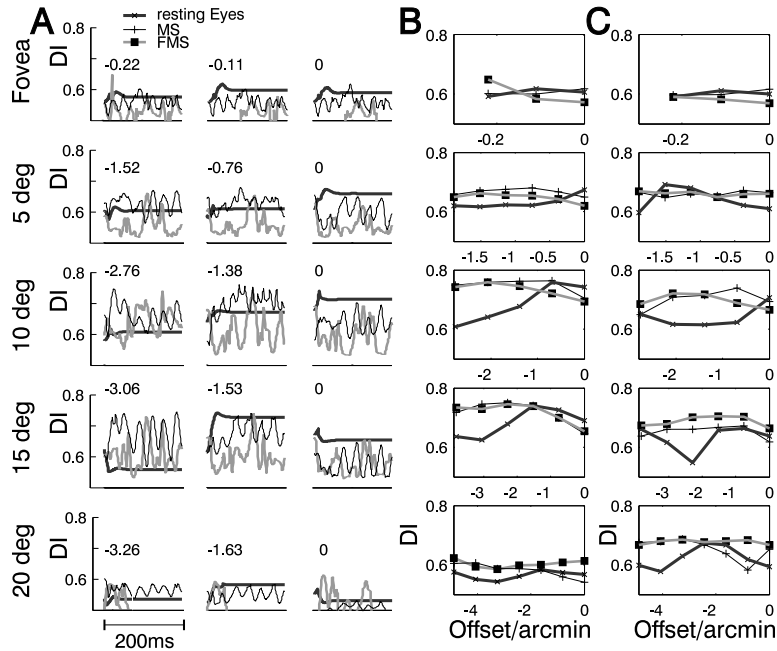

Figure 4: Temporal analysis of the ROC curves. A, Detectability index as function of time at different retinal eccentricities and different stimulus displacements relative to the ganglion cell positions (thick curves: resting eye, thin curves: slow+fast microsaccades+tremor, grey curves: fast microsaccades+tremor). Stimulus offsets are shown above the traces. B, Maximum of the curves in A at each eccentricity and location for the scaled PSF on a noisy ganglion cells grid. Only values are considered as a maximum where the $DI$ stays above the mean for $> 10ms$. C, Maximal $DI$ for the constant PSF.

now randomly oscillating across the smooth curves without micromovements. We note for most curves obtained with tremor there is an interval of at least 10ms where the $DI$ is substantially above its mean and equal or above the noise-free equivalent. Psychophysical evidence shows that detection tasks may require only short periods of as little as 5-10ms where the detectability must exceed threshold [17]. Thus in the retinal periphery the eye micromovements have a beneficial effect on the detectability by reducing aliasing.

In Fig. 4B, the maximum of $DI$ at different stimulus locations is plotted as function of the stimulus position. The maximum is defined as the largest value of the detectability index within a $> 10ms$ transient. The curves show the same effects as described above: Performance remains the similar in the central and improves in the peripheral retina. If the mean value of $DI$ instead of the maximum is considered, the effect is similar in the fovea, but no performance increase can be observed in the periphery (not shown). Fig. 4C shows the same analysis of responses for a constant PSF on a regular ganglion cell grid (see Fig. 1B), where aliasing occurs already at the photoreceptor level. The effect is very similar to that of the scaled PSF with stronger aliasing at higher eccentricities. However, at 10 and 15 deg, $DI$ is lower for all cases because the disarray of the ganglion cells allows for improved spatial averaging.

To summarize the previous results, the mean value of each curve in Fig. 4B and C is calculated. This can be interpreted as the psychophysical performance of a subject after many stimulus repetitions. They are shown in Fig. 5A for the scaled and Fig. 5B for the constant

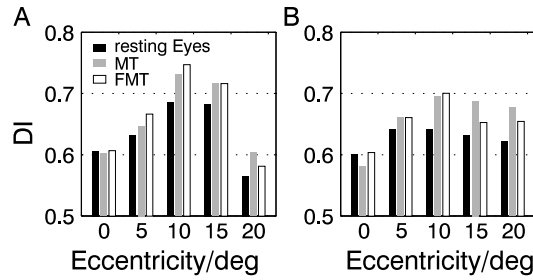

Figure 5: Mean detectability index ($DI$) for the experiments in Fig.3A (left, constant PSF) and B (right, PSF scaled proportional to cone-ganglion cell convergence ratio) as function of the retinal eccentricity.

PSF. The differences in $DI$ at different eccentricities is a result of the stimulus scaling. For both cases, eye micromovements increases the detectability at all eccentricities except in the fovea. For the two types of eye micromovements, the maximal relative improvement of $DI$ happens at different eccentricities. The first type, comprising microsaccades and tremor, frequently shifts the stimulus across adjoining ganglion cells at eccentricities $20°$. The second type has a smaller amplitude, thus the excitation of nearby ganglion cells is most efficient at $10°$. Thus, the effect depends on the spatial extend of the eye movements.

At $20°$, $DI$ is much lower for the scaled PSF on a noisy ganglion cell grid than for the constant PSF on the regular grid. Because $DI$ is consistently lower in the latter case for the other eccentricities, this indicates that here the effect of the spatial disarray can not be countered by spatial averaging of just four rows of ganglion cells.

Taken together, the results from the simulations shown here imply that a complex interplay of different factors affect the detectability of hyperacuity stimuli. Indeed the quantitative results from the model are very sensitive to changes of certain parameters (e.g. cell density). Equally, a great variability in human psychophysical performance exists. However, the effect of eye micromovements is consistent across the two cases shown here.

## 4   Discussion

Our results suggest that eye-micromovements contribute to visual hyperacuity in the peripheral visual field. By simulating ganglion cell responses for vernier stimuli using a realistic model and applying model-free ideal observer analysis, we show that in the retinal periphery eye-micromovements reduce the effect of aliasing due to neural undersampling. This leads to a higher detectability of hyperacuity stimuli. There has been a successful attempt to use small, continuous "scanning" movements to increase the resolution of a low resolution sensor array as a technical application [18]. We show that this principle can indeed be employed by vertebrates to improve visual acuity in certain (hyperacuity) tasks. However, eye movements have the reverse effect on detection tasks that require aliasing. Packer and Williams [19] have shown that in a high frequency (aliasing) grating detection task contrast thresholds are low for very brief and long presentation durations. For intermediate presentation times the threshold increases substantially. Because detection relies on aliasing, it requires a resting eye. This is more likely for very brief and long presentation times. For intermediate intervals, motion prevents aliasing. In hyperacuity, eye-micromovements increase detectability and we expect an asymptotic decrease of thresholds as function of the presentation time.

The question arises how eye-micromovements affect human psychophysical performance.

We predict an influence of the effect of stimulus presentation time for vernier targets between the central and peripheral retina. We would also expect an increase of detection thresholds under stabilized eye conditions in the periphey. This and further experiments also suggest that eye micromovements generally influence detection tasks that are performed close to the psychophysical threshold. It is further possible to directly apply the experimental procedure that was used in this work in an electrophysiological study. Specifically, it is possible to record from one ganglion cell with many different stimulus locations. These responses can then be used to reconstruct a spatial response profile equivalent to our simulated activity distribution (Fig.3B, F) and ROC analysis can be applied.

## References

[1] H.L. Averill and F.W. Weymouth. Visual perception and the retinal mosaic. II. The influence of eye-movements on the displacement threshold. *J Comp Psychol*, 5:147–176, 1925.

[2] W.H. Marshall and S.A. Talbot. Recent evidence for neural mechanisms in vision leading to a general theory of sensory acuity. *Biol Symp*, 7:117–164, 1942.

[3] R.M. Steinman and J.Z. Levinson. *Eye movements and their role in visual and cognitive processes*, chapter The role of eye movement in the detection of contrast and spatial detail, pages 115–212. Elsevier Science, 1990.

[4] M.H. Hennig, K. Funke, and F. Wörgötter. The influence of different retinal subcircuits on the nonlinearity of ganglion cell behavior. *J Neurosci*, 22:8726–8738, 2002.

[5] J. Sjöstrand, V. Olsson, Z. Popovic, and N. Conradi. Quantitative estimations of foveal and extra-foveal retinal circuitry in humans. *Vision Res*, 39:2987–2998, 1999.

[6] A.K. Goodchild, K.K. Ghosh, and P.R. Martin. Comparison of photoreceptor spatial density and ganglion cell morphology in the retina of human, macaque monkey, cat, and the marmoset callithrix jacchus. *J Comp Neurol*, 366:55–75, 1996.

[7] D.M. Dacey and M.R. Petersen. Dendritic field size and morphology of midget and parasol ganglion cells in the human retina. *Proc Natl Acad Sci USA*, 89:9666–9670, 1992.

[8] J.L. Schnapf, B.J. Nunn, M. Meister, and D.A. Baylor. Visual transduction in cones of the monkey macaca fascicularis. *J Physiol*, 427:681–713, 1990.

[9] D.M. Schneeweis and J.L. Schnapf. The photovoltage of marcaque cone photoreceptors: adapation, noise and kinetics. *J Neurosci*, 19(4):1203–1216, 1999.

[10] R.W. Rodieck and J. Stone. Analysis of receptive fields of cat retinal ganglion cells. *J Neurophysiol*, 28:833–849, 1965.

[11] L.J. Croner and E. Kaplan. Receptive fields of P and M ganglion cells across the primate retina. *Vision Res*, 35(1):7–24, 1995.

[12] G. Westheimer. *Handbook of Perception and Human Performance*, volume 1, chapter The eye as an optical instrument. John Wiley & Sons, New York, 1986.

[13] L.N. Thibos, D.L. Still, and Bradley A. Characterization of spatial aliasing and contrast sensitivity in peripheral vision. *Vision Res*, 36:249–58, 1996.

[14] M. Eizenman, P.E. Hallett, and R.C. Frecker. Power spectra for ocular drift and tremor. *Vision Res*, 25:1635–1640, 1985.

[15] D.H. Hubel and T.N. Wiesel. Receptive fields, binocular interaction, and functional architecture in the cat's visual cortex. *J Physiol*, 160:106–154, 1962.

[16] L.N. Thibos, D.J. Walsh, and Cheney F.E. Vision beyond the resolution limit: aliasing in the periphery. *Vision Res*, 27:2193–2197, 1987.

[17] A.B. Watson. *Handbook of perception and human performance*, volume 1, chapter Temporal sensitivity. Wiley, New York, 1986.

[18] Landolt O. and Mitros A. Visual sensor with resolution enhancement by mechanical vibrations. *Autonomous Robots*, 11:233–239, 2001.

[19] O. Packer and D.R. Williams. Blurring by fixational eye movements. *Vision Res*, 32:1931–1939, 1992.
